# Kernel Measures of Independence for non-*iid* Data

**Xinhua Zhang**
NICTA and Australian National University
Canberra, Australia
xinhua.zhang@anu.edu.au

**Le Song**[*]
School of Computer Science
Carnegie Mellon University, Pittsburgh, USA
lesong@cs.cmu.edu

**Arthur Gretton**
MPI Tübingen for Biological Cybernetics
Tübingen, Germany
arthur@tuebingen.mpg.de

**Alex Smola**[*]
Yahoo! Research
Santa Clara, CA, United States
alex@smola.org

## Abstract

Many machine learning algorithms can be formulated in the framework of statistical independence such as the Hilbert Schmidt Independence Criterion. In this paper, we extend this criterion to deal with structured and interdependent observations. This is achieved by modeling the structures using undirected graphical models and comparing the Hilbert space embeddings of distributions. We apply this new criterion to independent component analysis and sequence clustering.

## 1 Introduction

Statistical dependence measures have been proposed as a unifying framework to address many machine learning problems. For instance, clustering can be viewed as a problem where one strives to maximize the dependence between the observations and a discrete set of labels [14]. Conversely, if labels are given, feature selection can be achieved by finding a subset of features in the observations which maximize the dependence between labels and features [15]. Similarly in supervised dimensionality reduction [13], one looks for a low dimensional embedding which retains additional side information such as class labels. Likewise, blind source separation (BSS) tries to unmix independent sources, which requires a contrast function quantifying the dependence of the unmixed signals.

The use of mutual information is well established in this context, as it is theoretically well justified. Unfortunately, it typically involves density estimation or at least a nontrivial optimization procedure [11]. This problem can be averted by using the Hilbert Schmidt Independence Criterion (HSIC). The latter enjoys concentration of measure properties and it can be computed efficiently on any domain where a Reproducing Kernel Hilbert Space (RKHS) can be defined.

However, the application of HSIC is limited to independent and identically distributed (*iid*) data, a property that many problems do not share (*e.g.*, BSS on audio data). For instance many random variables have a pronounced temporal or spatial structure. A simple motivating example is given in Figure 1a. Assume that the observations $x_t$ are drawn *iid* from a uniform distribution on $\{0, 1\}$ and $y_t$ is determined by an XOR operation via $y_t = x_t \otimes x_{t-1}$. Algorithms which treat the observation pairs $\{(x_t, y_t)\}_{t=1}^{\infty}$ as *iid* will consider the random variables $x, y$ as independent. However, it is trivial to detect the XOR dependence by using the information that $x_i$ and $y_i$ are, in fact, sequences.

In view of its importance, temporal correlation has been exploited in the independence test for blind source separation. For instance, [9] used this insight to reject nontrivial nonseparability of nonlinear mixtures, and [18] exploited multiple time-lagged second-order correlations to decorrelate over time. These methods work well in practice. But they are rather *ad hoc* and appear very different from standard criteria. In this paper, we propose a framework which extends HSIC to structured non-*iid* data. Our new approach is built upon the connection between exponential family models and

---

[*]This work was partially done when the author was with the Statistical Machine Learning Group of NICTA.

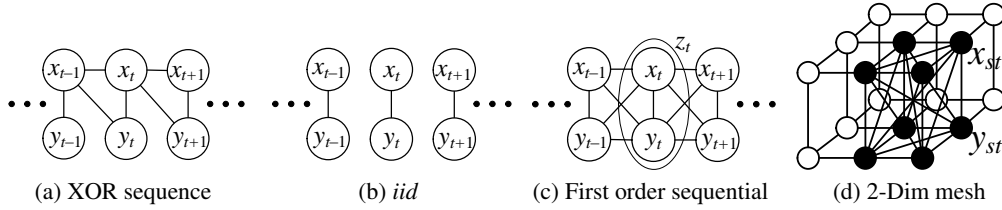

| (a) XOR sequence | (b) *iid* | (c) First order sequential | (d) 2-Dim mesh |

Figure 1: From left to right: (a) Graphical model representing the XOR sequence, (b) a graphical model representing *iid* observations, (c) a graphical model for first order sequential data, and (d) a graphical model for dependency on a two dimensional mesh.

the marginal polytope in an RKHS. This is doubly attractive since distributions can be uniquely identified by the expectation operator in the RKHS and moreover, for distributions with conditional independence properties the expectation operator decomposes according to the clique structure of the underlying undirected graphical model [2].

## 2 The Problem

Denote by $\mathcal{X}$ and $\mathcal{Y}$ domains from which we will be drawing observations $Z := \{(x_1, y_1), \ldots, (x_m, y_m)\}$ according to some distribution $p(x, y)$ on $\mathcal{Z} := \mathcal{X} \times \mathcal{Y}$. Note that the domains $\mathcal{X}$ and $\mathcal{Y}$ are fully general and we will discuss a number of different structural assumptions on them in Section 3 which allow us to recover existing and propose new measures of dependence. For instance $x$ and $y$ may represent sequences or a mesh for which we wish to establish dependence.

To assess whether $x$ and $y$ are independent we briefly review the notion of Hilbert Space embeddings of distributions [6]. Subsequently we discuss properties of the expectation operator in the case of conditionally independent random variables which will lead to a template for a dependence measure.

**Hilbert Space Embedding of Distribution**     Let $\mathcal{H}$ be a RKHS on $\mathcal{Z}$ with kernel $v : \mathcal{Z} \times \mathcal{Z} \mapsto \mathbb{R}$. Moreover, let $\mathcal{P}$ be the space of all distributions over $\mathcal{Z}$, and let $p \in \mathcal{P}$. The expectation operator in $\mathcal{H}$ and its corresponding empirical average can be defined as in [6]

$$\mu[p] := \mathbf{E}_{z \sim p(z)}[v(z, \cdot)] \qquad \text{such that} \qquad \mathbf{E}_{z \sim p(z)}[f(z)] = \langle \mu[p], f \rangle \qquad (1)$$

$$\mu[Z] := \frac{1}{m} \sum_{i=1}^{m} v((x^i, y^i), \cdot) \qquad \text{such that} \qquad \frac{1}{m} \sum_{i=1}^{m} f(x^i, y^i) = \langle \mu[Z], f \rangle. \qquad (2)$$

The map $\mu : \mathcal{P} \mapsto \mathcal{H}$ characterizes a distribution by an element in the RKHS. The following theorem shows that the map is injective [16] for a large class of kernels such as Gaussian and Laplacian RBF.

**Theorem 1** *If $\mathbf{E}_{z \sim p}[v(z, z)] < \infty$ and $\mathcal{H}$ is dense in the space of bounded continuous functions $C^0(\mathcal{Z})$ in the $L_\infty$ norm then the map $\mu$ is injective.*

### 2.1 Exponential Families

We are interested in the properties of $\mu[p]$ in the case where $p$ satisfies the conditional independence relations specified by an undirected graphical model. In [2], it is shown that for this case the sufficient statistics decompose along the maximal cliques of the conditional independence graph.

More formally, denote by $\mathcal{C}$ the set of maximal cliques of the graph $G$ and let $z_c$ be the restriction of $z \in \mathcal{Z}$ to the variables on clique $c \in \mathcal{C}$. Moreover, let $v_c$ be universal kernels in the sense of [17] acting on the restrictions of $\mathcal{Z}$ on clique $c \in \mathcal{C}$. In this case, [2] showed that

$$v(z, z') = \sum_{c \in \mathcal{C}} v_c(z_c, z'_c) \qquad (3)$$

can be used to describe all probability distributions with the above mentioned conditional independence relations using an exponential family model with $v$ as its kernel. Since for exponential families expectations of the sufficient statistics yield injections, we have the following result:

**Corollary 2** *On the class of probability distributions satisfying conditional independence properties according to a graph $G$ with maximal clique set $\mathcal{C}$ and with full support on their domain, the operator*

$$\mu[p] = \sum_{c \in \mathcal{C}} \mu_c[p_c] = \sum_{c \in \mathcal{C}} \mathbf{E}_{z_c}\left[v_c(z_c, \cdot)\right] \qquad (4)$$

*is injective if the kernels $v_c$ are all universal. The same decomposition holds for the empirical counterpart $\mu[Z]$.*

The condition of full support arises from the conditions of the Hammersley-Clifford Theorem [4, 8]: without it, not all conditionally independent random variables can be represented as the product of potential functions. Corollary 2 implies that we will be able to perform all subsequent operations on structured domains simply by dealing with mean operators on the corresponding maximal cliques.

## 2.2 Hilbert Schmidt Independence Criterion

Theorem 1 implies that we can quantify the difference between two distributions $p$ and $q$ by simply computing the square distance between their RKHS embeddings, *i.e.*, $\|\mu[p] - \mu[q]\|_{\mathcal{H}}^2$. Similarly, we can quantify the strength of dependence between random variables $x$ and $y$ by simply measuring the square distance between the RKHS embeddings of the joint distribution $p(x, y)$ and the product of the marginals $p(x) \cdot p(y)$ via

$$I(x, y) := \|\mu[p(x, y)] - \mu[p(x)p(y)]\|_{\mathcal{H}}^2 \,. \qquad (5)$$

Moreover, Corollary 2 implies that for an exponential family consistent with the conditional independence graph $G$ we may decompose $I(x, y)$ further into

$$I(x, y) = \sum_{c \in \mathcal{C}} \|\mu_c[p_c(x_c, y_c)] - \mu_c[p_c(x_c)p_c(y_c)]\|_{\mathcal{H}_c}^2$$
$$= \sum_{c \in \mathcal{C}} \left\{ \mathbf{E}_{(x_c y_c)(x_c' y_c')} + \mathbf{E}_{x_c y_c x_c' y_c'} - 2\mathbf{E}_{(x_c y_c)x_c' y_c'} \right\} \left[ v_c((x_c, y_c), (x_c', y_c')) \right] \qquad (6)$$

where bracketed random variables in the subscripts are drawn from their joint distributions and unbracketed ones are from their respective marginals, *e.g.*, $\mathbf{E}_{(x_c y_c)x_c' y_c'} := \mathbf{E}_{(x_c y_c)}\mathbf{E}_{x_c'}\mathbf{E}_{y_c'}$. Obviously the challenge is to find good empirical estimates of (6). In its simplest form we may replace each of the expectations by sums over samples, that is, by replacing

$$\mathbf{E}_{(x,y)}[f(x, y)] \leftarrow \frac{1}{m}\sum_{i=1}^{m} f(x_i, y_i) \quad \text{and} \quad \mathbf{E}_{(x)(y)}[f(x, y)] \leftarrow \frac{1}{m^2}\sum_{i,j=1}^{m} f(x_i, y_j). \qquad (7)$$

## 3 Estimates for Special Structures

To illustrate the versatility of our approach we apply our model to a number of graphical models ranging from independent random variables to meshes proceeding according to the following recipe:

1. Define a conditional independence graph.
2. Identify the maximal cliques.
3. Choose suitable joint kernels on the maximal cliques.
4. Exploit stationarity (if existent) in $I(x, y)$ in (6).
5. Derive the corresponding empirical estimators for each clique, and hence for all of $I(x, y)$.

## 3.1 Independent and Identically Distributed Data

As the simplest case, we first consider the graphical model in Figure 1b, where $\{(x_t, y_t)\}_{t=1}^{T}$ are *iid* random variables. Correspondingly the maximal cliques are $\{(x_t, y_t)\}_{t=1}^{T}$. We choose the joint kernel on the cliques to be

$$v_t((x_t, y_t), (x_t', y_t')) := k(x_t, x_t')l(y_t, y_t') \text{ hence } v((x, y), (x', y')) = \sum_{t=1}^{T} k(x_t, x_t')l(y_t, y_t'). \quad (8)$$

The representation for $v_t$ implies that we are taking an outer product between the Hilbert Spaces on $x_t$ and $y_t$ induced by kernels $k$ and $l$ respectively. If the pairs of random variables $(x_t, y_t)$ are *not* identically distributed, all that is left is to use (8) to obtain an empirical estimate via (7).

We may improve the estimate considerably if we are able to assume that all pairs $(x_t, y_t)$ are drawn from the *same* distribution $p(x_t, y_t)$. Consequently all coordinates of the mean

map are identical and we can use all the data to estimate just one of the discrepancies $\|\mu_c[p_c(x_c, y_c)] - \mu_c[p_c(x_c)p_c(y_c)]\|^2$. The latter expression is identical to the standard HSIC criterion and we obtain the biased estimate

$$\hat{I}(x, y) = \tfrac{1}{T} \operatorname{tr} HKHL \quad \text{where} \quad K_{st} := k(x_s, x_t), L_{st} := l(y_s, y_t) \text{ and } H_{st} := \delta_{st} - \tfrac{1}{T}. \quad (9)$$

## 3.2 Sequence Data

A more interesting application beyond *iid* data is sequences with a Markovian dependence as depicted in Figure 1c. Here the maximal cliques are the sets $\{(x_t, x_{t+1}, y_t, y_{t+1})\}_{t=1}^{T-1}$. More generally, for longer range dependency of order $\tau \in \mathbb{N}$, the maximal cliques will involve the random variables $(x_t, \ldots, x_{t+\tau}, y_t, \ldots, y_{t+\tau}) =: (x_{t,\tau}, y_{t,\tau})$.

We assume homogeneity and stationarity of the random variables: that is, all cliques share the same sufficient statistics (feature map) and their expected value is identical. In this case the kernel

$$v_0((x_{t,\tau}, y_{t,\tau}), (x'_{t,\tau}, y'_{t,\tau})) := k(x_{t,\tau}, x'_{t,\tau})l(y_{t,\tau}, y'_{t,\tau})$$

can be used to measure discrepancy between the random variables. Stationarity means that $\mu_c[p_c(x_c, y_c)]$ and $\mu_c[p_c(x_c)p_c(y_c)]$ are the same for all cliques $c$, hence $I(x, y)$ is a multiple of the difference for a single clique.

Using the same argument as in the *iid* case, we can obtain a biased estimate of the measure of dependence by using $K_{ij} = k(x_{i,\tau}, x_{j,\tau})$ and $L_{ij} = l(y_{i,\tau}, y_{j,\tau})$ instead of the definitions of $K$ and $L$ in (9). This works well in experiments. In order to obtain an unbiased estimate we need some more work. Recall the unbiased estimate of $I(x, y)$ is a fourth order U-statistic [6].

**Theorem 3** *An unbiased empirical estimator for* $\|\mu[p(x, y)] - \mu[p(x)p(y)]\|^2$ *is*

$$\hat{I}(x, y) := \tfrac{(m-4)!}{m!} \sum_{(i,j,q,r)} h(x_i, y_i, \ldots, x_r, y_r), \quad (10)$$

*where the sum is over all terms such that* $i, j, q, r$ *are mutually different, and*

$$h(x_1, y_1, \ldots, x_4, y_4) := \frac{1}{4!} \sum_{(t,u,v,w)}^{(1,2,3,4)} k(x_t, x_u)l(x_t, x_u) + k(x_t, x_u)l(x_v, x_w) - 2k(x_t, x_u)l(x_t, x_v),$$

*and the latter sum denotes all ordered quadruples* $(t, u, v, w)$ *drawn from* $(1, 2, 3, 4)$.

The theorem implies that in expectation $h$ takes on the value of the dependence measure. To establish that this also holds for dependent random variables we use a result from [1] which establishes convergence for stationary mixing sequences under mild regularity conditions, namely whenever the kernel of the U-statistic $h$ is bounded and the process generating the observations is absolutely regular. See also [5, Section 4].

**Theorem 4** *Whenever* $I(x, y) > 0$, *that is, whenever the random variables* are *dependent, the estimate* $\hat{I}(x, y)$ *is asymptotically normal with*

$$\sqrt{m}(\hat{I}(x, y) - I(x, y)) \xrightarrow{d} \mathcal{N}(0, 4\sigma^2) \quad (11)$$

*where the variance is given by*

$$\sigma^2 = \operatorname{Var}\left[h_3(x_1, y_1)\right]^2 + 2\sum_{t=1}^{\infty} \operatorname{Cov}(h_3(x_1, y_1), h_3(x_t, y_t)) \quad (12)$$

$$\text{and} \quad h_3(x_1, y_1) := \mathbf{E}_{(x_2, y_2, x_3, y_3, x_4, y_4)}[h(x_1, y_1, \ldots, x_4, y_4)] \quad (13)$$

This follows from [5, Theorem 7], again under mild regularity conditions (note that [5] state their results for U-statistics of second order, and claim the results hold for higher orders). The proof is tedious but does not require additional techniques and is therefore omitted.

## 3.3 TD-SEP as a special case

So far we did not discuss the freedom of choosing different kernels. In general, an RBF kernel will lead to an effective criterion for measuring the dependence between random variables, especially in

time-series applications. However, we could also choose linear kernels for $k$ and $l$, for instance, to obtain computational savings.

For a specific choice of cliques and kernels, we can recover the work of [18] as a special case of our framework. In [18], for two centered scalar time series $x$ and $y$, the contrast function is chosen as the sum of same-time and time-lagged cross-covariance $\mathbf{E}[x_t y_t] + \mathbf{E}[x_t y_{t+\tau}]$. Using our framework, two types of cliques, $(x_t, y_t)$ and $(x_t, y_{t+\tau})$, are considered in the corresponding graphical model. Furthermore, we use a joint kernel of the form

$$\langle x_s, x_t \rangle \langle y_s, y_t \rangle + \langle x_s, x_t \rangle \langle y_{s+\tau}, y_{t+\tau} \rangle, \tag{14}$$

which leads to the estimator of structured HSIC: $I(x, y) = \frac{1}{T} \left( \operatorname{tr} HKHL + \operatorname{tr} HKHL_\tau \right)$. Here $L_\tau$ denotes the linear covariance matrix for the time lagged $y$ signals. For scalar time series, basic algebra shows that $\operatorname{tr} HKHL$ and $\operatorname{tr} HKHL_\tau$ are the estimators of $\mathbf{E}[x_t y_t]$ and $\mathbf{E}[x_t y_{t+\tau}]$ respectively (up to a multiplicative constant).

Further generalization can incorporate several time lagged cross-covariances into the contrast function. For instance, TD-SEP [18] uses a range of time lags from 1 to $\tau$. That said, by using a nonlinear kernel we are able to obtain better contrast functions, as we will show in our experiments.

### 3.4 Grid Structured Data

Structured HSIC can go beyond sequence data and be applied to more general dependence structures such as 2-D grids for images. Figure 1d shows the corresponding graphical model. Here each node of the graphical model is indexed by two subscripts, $i$ for row and $j$ for column. In the simplest case, the maximal cliques are

$$\mathcal{C} = \{(x_{ij}, x_{i+1,j}, x_{i,j+1}, x_{i+1,j+1}, y_{ij}, y_{i+1,j}, y_{i,j+1}, y_{i+1,j+1})\}_{ij}.$$

In other words, we are using a cross-shaped stencil to connect vertices. Provided that the kernel $v$ can also be decomposed into the product of $k$ and $l$, then a biased estimate of the independence measure can be again formulated as $\operatorname{tr} HKHL$ up to a multiplicative constant. The statistical analysis of U-statistics for stationary Markov random fields is highly nontrivial. We are not aware of results equivalent to those discussed in Section 3.2.

## 4 Experiments

Having a dependence measure for structured spaces is useful for a range of applications. Analogous to *iid* HSIC, structured HSIC can be applied to non-*iid* data in applications such as independent component analysis [12], independence test [6], feature selection [15], clustering [14], and dimensionality reduction [13]. The fact that structured HSIC can take into account the interdependency between observations provides us with a principled generalization of these algorithms to, *e.g.*, time series analysis. In this paper, we will focus on two examples: independent component analysis, where we wish to minimize the dependence, and time series segmentation, where we wish to maximize the dependence instead. Two simple illustrative experiments on independence test for XOR binary sequence and Gaussian process can be found in the longer version of this paper.

### 4.1 Independent Component Analysis

In independent component analysis (ICA), we observe a time series of vectors $u$ that corresponds to a linear mixture $u = As$ of $n$ mutually independent sources $s$ (each entry in the source vector here is a random process, and depends on its past values; examples include music and EEG time series). Based on the series of observations $t$, we wish to recover the sources using only the independence assumption on $s$. Note that sources can only be recovered up to scaling and permutation. The core of ICA is a contrast function that measures the independence of the estimated sources. An ICA algorithm searches over the space of mixing matrix $A$ such that this contrast function is minimized. Thus, we propose to use structured HSIC as the contrast function for ICA. By incorporating time lagged variables in the cliques, we expect that structured HSIC can better deal with the non-*iid* nature of time series. In this respect, we generalize the TD-SEP algorithm [18], which implements this idea using a linear kernel on the signal. Thus, we address the question of whether correlations between higher order moments, as encoded using non-linear kernels, can improve the performance of TD-SEP on real data.

Table 1: Median performance of ICA on music using HSIC, TDSEP, and structured HSIC. In the top row, the number $n$ of sources and $m$ of samples are given. In the second row, the number of time lags $\tau$ used by TDSEP and structured HSIC are given: thus the observation vectors $x, x_{t-1}, \ldots, x_{t-\tau}$ were compared. The remaining rows contain the median Amari divergence (multiplied by 100) for the three methods tested. The original HSIC method does not take into account time dependence ($\tau = 0$), and returns a single performance number. Results are in all cases averaged over 136 repetitions: for two sources, this represents all possible pairings, whereas for larger $n$ the sources are chosen at random without replacement.

| Method | $n = 2, m = 5000$ | | | $n = 3, m = 10000$ | | | $n = 4, m = 10000$ | | |
|---|---|---|---|---|---|---|---|---|---|
| | 1 | 2 | 3 | 1 | 2 | 3 | 1 | 2 | 3 |
| HSIC | 1.51 | | | 1.70 | | | 2.68 | | |
| TDSEP | 1.54 | 1.62 | 1.74 | 1.84 | 1.72 | 1.54 | 2.90 | 2.08 | 1.91 |
| Structured HSIC | 1.48 | 1.62 | 1.64 | 1.65 | 1.58 | 1.56 | 2.65 | 2.12 | 1.83 |

**Data** Following the settings of [7, Section 5.5], we unmixed various musical sources, combined using a randomly generated orthogonal matrix $A$ (since optimization over the orthogonal part of a general mixing matrix is the more difficult step in ICA). We considered mixtures of two to four sources, drawn at random without replacement from 17 possibilities. We used the sum of pairwise dependencies as the overall contrast function when more than two sources were present.

**Methods** We compared structured HSIC to TD-SEP and *iid* HSIC. While *iid* HSIC does not take the temporal dependence in the signal into account, it has been shown to perform very well for *iid* data [12]. Following [7], we employed a Laplace kernel, $k(x, x') = \exp(-\lambda\|x - x'\|)$ with $\lambda = 3$ for both structured and *iid* HSIC. For both structured and *iid* HSIC, we used gradient descent over the orthogonal group with a Golden search, and low rank Cholesky decompositions of the Gram matrices to reduce computational cost, as in [3].

**Results** We chose the Amari divergence as the index for comparing performance of the various ICA methods. This is a divergence measure between the estimated and true unmixing matrices, which is invariant to the output ordering and scaling ambiguities. A smaller Amari divergence indicates better performance. Results are shown in Table 1. Overall, contrast functions that take time delayed information into account perform best, although the best time lag is different when the number of sources varies.

## 4.2 Time Series Clustering and Segmentation

We can also extend clustering to time series and sequences using structured HSIC. This is carried out in a similar way to the *iid* case. One can formulate clustering as generating the labels $y$ from a finite discrete set, such that their dependence on $x$ is maximized [14]:

$$\text{maximize}_y \ \text{tr} \, HKHL \quad \textit{subject to} \text{ constraints on } y. \tag{15}$$

Here $K$ and $L$ are the kernel matrices for $x$ and the generated $y$ respectively. More specifically, assuming $L_{st} := \delta(y_s, y_t)$ for discrete labels $y$, we recover clustering. Relaxing discrete labels to $y_t \in \mathbb{R}$ with bounded norm $\|y\|_2$ and setting $L_{st} := y_s y_t$, we obtain Principal Component Analysis.

This reasoning for *iid* data carries over to sequences by introducing additional dependence structure through the kernels: $K_{st} := k(x_{s,\tau}, x_{t,\tau})$ and $L_{st} := l(y_{s,\tau}, y_{t,\tau})$. In general, the interacting label sequences make the optimization in (15) intractable. However, for a class of kernels $l$ an efficient decomposition can be found by applying a reverse convolution on $k$: assume that $l$ is given by

$$l(y_{s,\tau}, y_{t,\tau}) = \sum\nolimits_{u,v=0}^{\tau} \bar{l}(y_{s+u}, y_{t+v}) M_{uv}, \tag{16}$$

where $M \in \mathbb{R}^{(\tau+1)\times(\tau+1)}$ with $M \succeq 0$, and $\bar{l}$ is a base kernel between individual time points. A common choice is $\bar{l}(y_s, y_t) = \delta(y_s, y_t)$. In this case we can rewrite $\text{tr} \, HKHL$ by applying the summation over $M$ to $HKH$, *i.e.*,

$$\sum_{s,t=1}^{T} [HKH]_{ij} \sum_{u,v=0}^{\tau} \bar{l}(y_{s+u}, y_{t+v}) M_{uv} = \sum_{s,t=1}^{T+\tau} \underbrace{\sum_{\substack{u,v=0 \\ s-u, t-v \in [1,T]}}^{\tau} M_{uv} [HKH]_{s-u, t-v}}_{:= \bar{K}_{st}} \bar{l}(y_s, y_t) \tag{17}$$

Table 2: Segmentation errors by various methods on the four studied time series.

| Method | Swimming I | Swimming II | Swimming II | BCI |
|---|---|---|---|---|
| structured HSIC | **99.0** | **118.5** | **108.6** | **111.5** |
| spectral clustering | 125 | 212.3 | 143.9 | 162 |
| HMM | 153.2 | 120 | 150 | 168 |

This means that we may apply the matrix $M$ to $HKH$ and thereby we are able to decouple the dependency within $y$. Denote the convolution by $\bar{K} = [HKH] \star M$. Consequently using $\bar{K}$ we can directly apply (15) to times series and sequence data. In practice, approximate algorithms such as incomplete Cholesky decomposition are needed to efficiently compute $\bar{K}$.

**Datasets** We study two datasets in this experiment. The first dataset is collected by the Australian Institute of Sports (AIS) from a 3-channel orientation sensor attached to a swimmer. The three time series we used in our experiment have the following configurations: $T = 23000$ time steps with 4 laps; $T = 47000$ time steps with 16 laps; and $T = 67000$ time steps with 20 laps. The task is to automatically find the starting and finishing time of each lap based on the sensor signals. We treated this problem as a segmentation problem. Since the dataset contains 4 different style of swimming, we used 6 as the number of clusters (there are 2 additional clusters for starting and finishing a lap).

The second dataset is a brain-computer interface data (data IVb of Berlin BCI group[1]). It contains EEG signals collected when a subject was performing three types of cued imagination. Furthermore, the relaxation period between two imagination is also recorded in the EEG. Including the relaxation period, the dataset consists of $T = 10000$ time points with 16 different segments. The task is to automatically detect the start and end of an imagination. We used 4 clusters for this problem.

**Methods** We compared three algorithms: structured HSIC for clustering, spectral clustering [10], and HMM. For structured HSIC, we used the maximal cliques of $(x_t, y_{t-50,100})$, where $y$ is the discrete label sequence to be generated. The kernel $l$ on $y$ took the form of equation (16), with $M \in \mathbb{R}^{101 \times 101}$ and $M_{uv} := \exp(-(u-v)^2)$. The kernel $k$ on $x$ was Gaussian RBF: $\exp(-\|x - x'\|^2)$. As a baseline, we used a spectral clustering with the same kernel $k$ on $x$, and a first order HMM with 6 hidden states and diagonal Gaussian observation model[2].

Further details regarding preprocessing of the above two datasets (which is common to all algorithms subsequently compared), parameters of algorithms and protocols of experiments, are available in the longer version of this paper.

**Results** To evaluate the segmentation quality, the boundaries found by various methods were compared to the ground truth. First, each detected boundary was matched to a true boundary, and then the discrepancy between them was counted into the error. The overall error was this sum divided by the number of boundaries. Figure 2d gives an example on how to compute this error.

According to Table 2, in all of the four time series we studied, segmentation using structured HSIC leads to lower error compared with spectral clustering and HMM. For instance, structured HSIC reduces nearly $1/3$ of the segmentation error in the BCI dataset. To provide a visual feel of the improvement, we plot the true boundaries together with the segmentation results in Figure 2a, 2b,2c. Clearly, segment boundaries produced by structured HSIC fit better with the ground truth.

## 5 Conclusion

In this paper, we extended the Hilbert Schmidt Independence Criterion from *iid* data to structured and non-*iid* data. Our approach is based on RKHS embeddings of distributions, and utilizes the efficient factorizations provided by the exponential family associated with undirected graphical models. Encouraging experimental results were demonstrated on independence test, ICA, and segmentation for time series. Further work will be done in the direction of applying structured HSIC to PCA and feature selection on structured data.

**Acknowledgements**

NICTA is funded by the Australian Governments Backing Australia's Ability and the Centre of Excellence programs. This work is also supported by the IST Program of the European Community, under the FP7 Network of Excellence, ICT-216886-NOE.

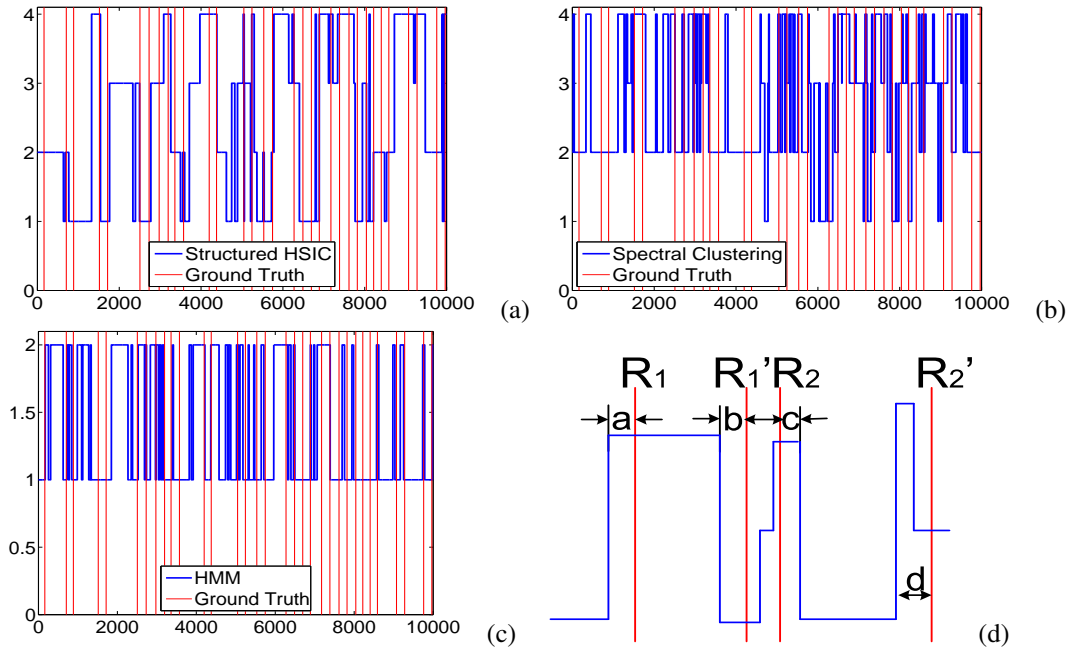

Figure 2: Segmentation results produced by (a) structured HSIC, (b) spectral clustering and (c) HMM. (d) An example for counting the segmentation error. Red line denotes the ground truth and blue line is the segmentation results. The error introduced for segment $R_1$ to $R_1'$ is $a + b$, while that for segment $R_2$ to $R_2'$ is $c + d$. The overall error in this example is then $(a + b + c + d)/4$.

## Footnotes

[1]http://ida.first.fraunhofer.de/projects/bci/competition-iii/desc-IVb.html

[2]http://www.torch.ch

# References

[1] Aaronson, J., Burton, R., Dehling, H., Gilat, D., Hill, T., & Weiss, B. (1996). Strong laws for L and U-statistics. *Transactions of the American Mathematical Society*, *348*, 2845–2865.

[2] Altun, Y., Smola, A. J., & Hofmann, T. (2004). Exponential families for conditional random fields. In *UAI*.

[3] Bach, F. R., & Jordan, M. I. (2002). Kernel independent component analysis. *JMLR*, *3*, 1–48.

[4] Besag, J. (1974). Spatial interaction and the statistical analysis of lattice systems (with discussion). *J. Roy. Stat. Soc. B*, *36*(B), 192–326.

[5] Borovkova, S., Burton, R., & Dehling, H. (2001). Limit theorems for functionals of mixing processes with applications to dimension estimation. *Transactions of the American Mathematical Society*, *353*(11), 4261–4318.

[6] Gretton, A., Fukumizu, K., Teo, C.-H., Song, L., Schölkopf, B., & Smola, A. (2008). A kernel statistical test of independence. Tech. Rep. 168, MPI for Biological Cybernetics.

[7] Gretton, A., Herbrich, R., Smola, A., Bousquet, O., & Schölkopf, B. (2005). Kernel methods for measuring independence. *JMLR*, *6*, 2075–2129.

[8] Hammersley, J. M., & Clifford, P. E. (1971). Markov fields on finite graphs and lattices. Unpublished manuscript.

[9] Hosseni, S., & Jutten, C. (2003). On the separability of nonlinear mixtures of temporally correlated sources. *IEEE Signal Processing Letters*, *10*(2), 43–46.

[10] Ng, A., Jordan, M., & Weiss, Y. (2002). On spectral clustering: Analysis and an algorithm. In *NIPS*.

[11] Nguyen, X., Wainwright, M. J., & Jordan, M. I. (2008). Estimating divergence functionals and the likelihood ratio by penalized convex risk minimization. In *NIPS*.

[12] Shen, H., Jegelka, S., & Gretton, A. (submitted). Fast kernel-based independent component analysis. *IEEE Transactions on Signal Processing*.

[13] Song, L., Smola, A., Borgwardt, K., & Gretton, A. (2007). Colored maximum variance unfolding. In *NIPS*.

[14] Song, L., Smola, A., Gretton, A., & Borgwardt, K. (2007). A dependence maximization view of clustering. In *Proc. Intl. Conf. Machine Learning*.

[15] Song, L., Smola, A., Gretton, A., Borgwardt, K., & Bedo, J. (2007). Supervised feature selection via dependence estimation. In *ICML*.

[16] Sriperumbudur, B., Gretton, A., Fukumizu, K., Lanckriet, G., & Schölkopf, B. (2008). Injective hilbert space embeddings of probability measures. In *COLT*.

[17] Steinwart, I. (2002). The influence of the kernel on the consistency of support vector machines. *JMLR*, *2*.

[18] Ziehe, A., & Müller, K.-R. (1998). TDSEP – an efficient algorithm for blind separation using time structure. In *ICANN*.
